# A Novel Channel Selection System in Cochlear Implants Using Artificial Neural Network

**Marwan A. Jabri**    &    **Raymond J. Wang**
Systems Engineering and Design Automation Laboratory
Department of Electrical Engineering
The University of Sydney
NSW 2006, Australia
{marwan,jwwang}@sedal.usyd.edu.au

## Abstract

State-of-the-art speech processors in cochlear implants perform channel selection using a spectral maxima strategy. This strategy can lead to confusions when high frequency features are needed to discriminate between sounds. We present in this paper a novel channel selection strategy based upon pattern recognition which allows "smart" channel selections to be made. The proposed strategy is implemented using multi-layer perceptrons trained on a multi-speaker labelled speech database. The input to the network are the energy coefficients of $N$ energy channels. The output of the system are the indices of the $M$ selected channels.

We compare the performance of our proposed system to that of spectral maxima strategy, and show that our strategy can produce significantly better results.

## 1 INTRODUCTION

A cochlear implant is a device used to provide the sensation of sound to those who are profoundly deaf by means of electrical stimulation of residual auditory neurons. It generally consists of a directional microphone, a wearable speech processor, a head-set transmitter and an implanted receiver-stimulator module with an electrode

array which all together provide an electrical representation of the speech signal to the residual nerve fibres of the peripheral auditory system (Clark *et al*, 1990).

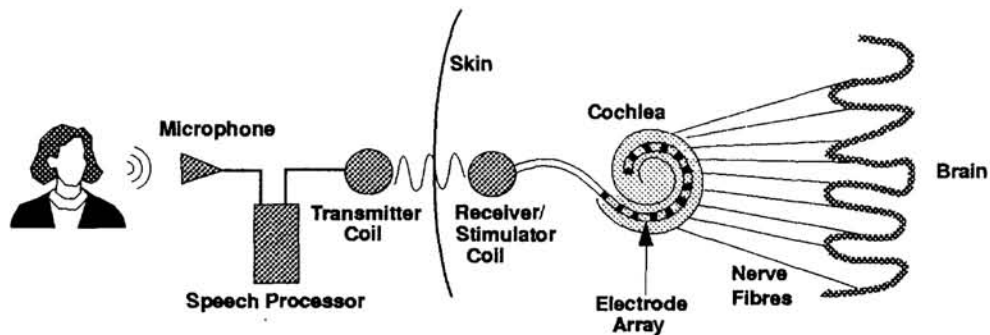

Figure 1: A simplified schematic diagram of the cochlear implants

A simplified schematic diagram of the cochlear implants is shown in Figure 1. Speech sounds are picked up by the directional microphone and sent to the speech processor. The speech processor amplifies, filters and digitizes these signals, and then selects and codes the appropriate sound information. The coded signal contains information as to which electrode to stimulate and the intensity level required to generate the appropriate sound sensations. The signal is then sent to the receiver/stimulator via the transmitter coil. The receiver/stimulator delivers electrical impulses to the appropriate electrodes in the cochlea. These stimulated electrodes then directly activate the hearing nerve in the inner ear, creating the sensation of sound, which is then forwarded to the brain for interpretation. The entire process happens in milliseconds.

For multi-channel cochlear implants, the task of the speech processor is to compute the spectral energy of the electrical signals it receives, and to quantise them into different levels. The energy spectrum is commonly divided into separate bands using a filter bank of $N$ (typically 20) bandpass filters with centre frequencies ranging from 250 Hz to 10 KHz. The bands of energy are allocated to electrodes in the patient's implant on a one-to-one basis. Usually the most-apical bipolar electrode pairs are allocated to the channels in tonotopic order. The limitations of implant systems usually require only a selected number of the quantised energy levels to be fed to the implanted electrode array (Abbas, 1993; Schouten, 1992).

The state-of-the-art speech processor for multi-channel implants performs channel selection using spectral maxima strategy (McDermott *et al*, 1992; Seligman & McDermott, 1994). The maxima strategy selects the $M$ (about 6) largest spectral energy of the frequency spectrum as stimulation channels from a filter bank of $N$ (typically 20) bandpass. It is believed that compared to other channel selection techniques (F0F2, F0F1F2, MPEAK ... ), the maxima strategy increases the amount of spectral information and improves the speech perception and recognition performance.

However, maxima strategy relies heavily on the highest energies. This often leads to the same levels being selected for different sounds, as the energy levels that distinguish them are not high enough to be selected. For some speech signals,

it does not cater for confusions and cannot discriminate between high frequency features.

We present in this paper Artificial Neural Networks (ANN) techniques for implementing "smart" channel selection for cochlear implant systems. The input to the proposed channel selection system consists of the energy coefficients (18 in our experiments) and the output the indices of the selected channels (6 in our experiments). The neural network based selection system is trained on a multi-speaker labelled speech and has been evaluated on a separated multi-speaker database not used in the training phase. The most important feature of our ANN based channel selection system is its ability to select the channels for stimulation on the basis of the overall morphology of the energy spectrum and not only on the basis of the maximal energy values.

## 2   THE PATTERN RECOGNITION BASED CHANNEL SELECTION STRATEGY

Speech is the most natural form of human communication. The speech information signal can be divided into phonemes, which share some common acoustic properties with one another for a short interval of time. The phonemes are typically divided into two broad classes: (a) vowels, which allow unrestricted airflow in the vocal tract, and (b) consonants, which restrict airflow at some point and are weaker than vowels. Different phonemes have different morphology in the energy spectrum. Moreover, for different speakers and different speech sentences, the same phonemes have different energy spectrum morphologies (Kent & Read, 1992). Therefore, simple methods to select some of the most important channels for all the phoneme patterns will not perform as good as the method that considers the spectrum in its entirety.

The existing maxima strategy only refers to the spectrum amplitudes found in the entire estimated spectrum without considering the morphology. Typically several of the maxima results can be obtained from a single spectral peak. Therefore, for some phoneme patterns, the selection result is good enough to represent the original phoneme. But for some others, some important features of the phoneme are lost. This usually happens to those phonemes with important features in the high frequency region. Due to the low amplitude of the high frequency in the spectrum morphology, maxima methods are not capable to extract those high frequency features. The relationship between the desired $M$ output channels and the energy spectrum patterns is complex, and depending on the conditions, may be influenced by many factors. As mentioned in the Introduction, channel selection methods that make use of local information only in the energy spectrum are bound to produce channel sub-sets where sounds may be confused. The confusions can be reduced if "global" information of the energy spectrum is used in the selection process.

The channel selection approach we are proposing makes use of the overall energy spectrum. This is achieved by turning the selection problem into that of a spectrum morphology pattern recognition one and hence, we call our approach Pattern Recognition based Channel Selection (PRCS).

## 2.1  PRCS STRATEGY

The PRCS strategy is implemented using two cascaded neural networks shown in Figure 2:

- Spectral morphological classifier: Its inputs are the spectrum energy amplitudes of all the channels and its outputs all the transformations of the inputs. The transformation between input and output can be seen as a recognition, emphasis, and/or decaying of the inputs. The consequence is that some inputs are amplified and some decayed, depending on the morphology of the spectrum. The classifier performs a non-linear mapping.

- $M$ strongest of $N$ classifier: It receives the output of morphological classifier and applies a $M$ strongest selection rule.

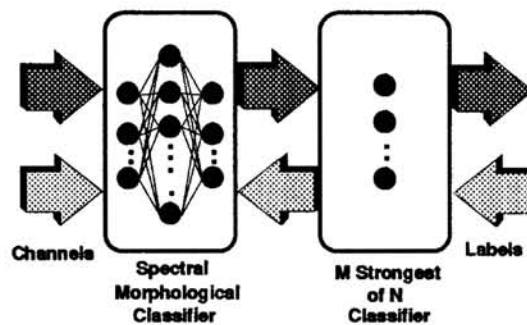

Figure 2: The pattern recognition based channel selection architecture

## 2.2  TRAINING AND TESTING DATA

The most difficult task in developing the proposed PRCS is to set up the labelled training and testing data for the spectral morphological classifier.

The training and testing data sets have been constructed using the process shown in Figure 3.

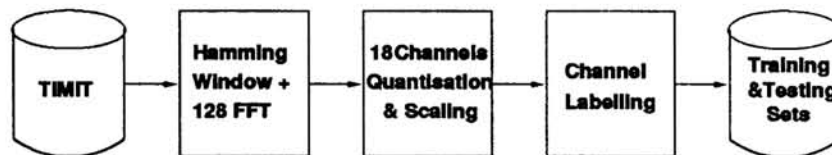

Figure 3: The process of generating training and testing sets

The sounds in the data sets are speech extracted from the DARPA TIMIT multi-speaker speech corpus (Fisher *et al*, 1987) which contains a total of 6300 sentences, 10 sentences spoken by each of 630 speakers. The speech signal is sampled at 16KHz rate with 16 bit precision. As the speech is nonstationary, to produce the energy spectrum versus channel numbers, a short-time speech analysis method is used.

The Fast Fourier Transform with $8ms$ smooth Hamming window technique is applied to yield the energy spectrum. The hamming window has the shape of a raised

cosine pulse:

$$h(n) = \begin{cases} 0.54 - 0.46\cos\left(\frac{2\pi n}{N-a}\right) & \text{for } 0 \leq n \leq N - 1 \\ 0 & \text{otherwise} \end{cases}$$

The time frame on which the speech analysis is performed is $4ms$ long and the successive time frame windows overlap by 50%.

Using frequency allocations similar to that used in commercial cochlear implant speech processors, the frequency range in the spectrum is divided into 18 channels with each channel having the center frequencies of *250, 450, 650, 850 1050, 1250, 1450, 1650, 1895, 2177, 2500, 2873, 3300, 3866, 4530, 5307, 6218* and *7285Hz* respectively. Each energy spectrum from a time frame is quantised into these 18 frequency bands. The energy amplitude for each level is the sum of the amplitude value of the energy for all the frequency components in the level.

The quantised energy spectrum is then labelled using a graphics based tool, called LABEL, developed specially for this application. LABEL displays the spectrum pattern including the unquantised spectrum, the signal source, speaker's name, speech sentence, phoneme, signal pre-processing method and FFT results. All these information assists labelling experts to allocate a score (1 to 18) to each channel. The score reflects the importance of the information provided by each of the bands. Hence, if six channels are only to be selected, the channels with the score 1 to 6 can be used and are highlighted. The labelling is necessary as a supervised neural network training method is being used.

A total of 5000 energy spectrum patterns have been labelled. They are from 20 different speakers and different spoken sentences. Of the 5000 example patterns, 4000 patterns are allocated for training and 1000 patterns for testing.

## 3  EXPERIMENTAL RESULTS

We have implemented and tested the PRCS system as described above and our experiments show that it has better performance than channel selection systems used in present cochlear implant processors.

The PRCS system is effectively constructed as a multi-module neural network using MUME (Jabri *et al*, 1994). The back-propagation algorithm in an on-line mode is used to train the MLP. The training patterns input components are the energy amplitudes of the 18 channels and the teacher component consists of a "1" for a channel to be selected and "0" for all others. The MLP is trained for up to 2000 epochs or when a minimum total mean squared error is reached. A learning rate $\eta$ of 0.01 is used (no weight decay).

We show the average performance of our PRCS in Table 1 where we also show the performance of a leading commercial spectral maxima strategy called SPEAK on the same test set. In the first column of this table we show the number of channels that matched out of the 6 desired channels. For example, the first row corresponds to the case where all 6 channels match the desired 6 channels in the test data base, and so on. As Table 1 shows, the PRCS produces a significantly better performance than the commercial strategy on the speech test set.

The selection performance to different phonemes is listed in Table 2. It clearly

Table 1: The comparison of average performance between commercial and PRCS system

| The Channel Selections from the two different methods | | |
|---|---|---|
| | PRCS results | Commercial technique results |
| Fully matched | 22 % | 4 % |
| 5 matched | 80 % | 25 % |
| 4 matched | 98 % | 57 % |
| 3 matched | 100 % | 93 % |
| 2 matched | 100 % | 99 % |
| 1 matched | 100 % | 100 % |

Table 2: PRCS channel selecting performance on different phoneme patterns

| The PRCS results for different phoneme patterns | | | | |
|---|---|---|---|---|
| Phoneme | Fully matched | 5 matched | 4 matched | 3 matched |
| Stops | 19 % | 69 % | 96 % | 100 % |
| Fricatives | 18 % | 66 % | 92 % | 100 % |
| Nasals | 14 % | 66 % | 96 % | 100 % |
| Semivowels & Glides | 14 % | 79 % | 95 % | 100 % |
| Vowels | 25 % | 84 % | 98 % | 100 % |

shows that the PRCS strategy can cater for the features of all the speech spectrum patterns.

To compare the practical performance of the PRCS with the maxima strategies we have developed a direct performance test system which allows us to play the synthesized speech of the selected channels through post-speech synthesizer. Our test shows that the PRCS produces more intelligible speech to the normal ears. Sixteen different sentences spoken by sixteen people are tested using both maxima and PRCS methods. It is found that the synthesized speech from PRCS has much more high frequency features than that of the speech produced by the maxima strategy. All listeners who were asked to take the test agreed that the quality of the speech sound from PRCS is much better than those from the commercial maxima channel selection system. The tape recording of the synthesized speech will be available at the conference.

## 4    CONCLUSION

A pattern recognition based channel selection strategy for Cochlear Implants has been presented. The strategy is based on a 18-72-18 MLP strongest selector. The proposed channel selection strategy has been compared to a leading commercial technique. Our simulation and play back results show that our machine learning based technique produces significantly better channel selections.

## Reference

Abbas, P. J. (1993) Electrophysiology. *"Cochlear Implants: Audiological Foundations" edited by R. S. Tyler, Singular Publishing Group*, pp.317–355.

Clark, G. M., Tong, Y. C.& Patrick, J. F. (1990) Cochlear Prosthesis. *Edinborough: Churchill Living stone.*

Fisher, W. M., Zue, V., Bernstein, J. & Pallett, D. (1987) An Acoustic-Phonetic Data Base. *In 113th Meeting of Acoust Soc Am*, May 1987

Jabri, M. A., Tinker, E. A. & Leerink, L. (1994) MUME — A Multi-Net Multi-Architecture Neural Simulation Environment. *"Neural Network Simulation Environments", J. Skrzypek ed.*, Kluwer Academic Publishers.

Kent, R. D. & Read, C. (1992) The Acoustic Analysis of Speech. *Whurr Publishers.*

McDermott, H. J., McKay, C. M. & Vandali, A. E. (1992) A new portable sound processor for the University of Melbourne / Nucleus Limited multielectrode cochlear implant. *J. Acoust. Soc. Am.* 91(6), June 1992, pp.3367-3371

Schouten, M. E. H edited (1992) The Auditory Processing of Speech — From Sounds to Words. *Speech Research 10, Mouton de Gruyter.*

Seligman, P. & McDermott, H. (1994) Architecture of the SPECTRA 22 Speech Processor. *International Cochlear Implant, Speech and Hearing Symposium*, Melbourne, October, 1994, p.254.
